# Batch Value Function Approximation via Support Vectors

**Thomas G Dietterich**
Department of Computer Science
Oregon State University
Corvallis, OR, 97331
*tgd@cs.orst.edu*

**Xin Wang**
Department of Computer Science
Oregon State University
Corvallis, OR, 97331
*wangxi@cs.orst.edu*

## Abstract

We present three ways of combining linear programming with the kernel trick to find value function approximations for reinforcement learning. One formulation is based on SVM regression; the second is based on the Bellman equation; and the third seeks only to ensure that good moves have an advantage over bad moves. All formulations attempt to minimize the number of support vectors while fitting the data. Experiments in a difficult, synthetic maze problem show that all three formulations give excellent performance, but the advantage formulation is much easier to train. Unlike policy gradient methods, the kernel methods described here can easily adjust the complexity of the function approximator to fit the complexity of the value function.

## 1 Introduction

Virtually all existing work on value function approximation and policy-gradient methods starts with a parameterized formula for the value function or policy and then seeks to find the best policy that can be represented in that parameterized form. This can give rise to very difficult search problems for which the Bellman equation is of little or no use. In this paper, we take a different approach: rather than fixing the form of the function approximator and searching for a representable policy, we instead identify a good policy and then search for a function approximator that can represent it. Our approach exploits the ability of mathematical programming to represent a variety of constraints including those that derive from supervised learning, from advantage learning (Baird, 1993), and from the Bellman equation. By combining the kernel trick with mathematical programming, we obtain a function approximator that seeks to find the smallest number of support vectors sufficient to represent the desired policy. This side-steps the difficult problem of searching for a good policy among those policies representable by a fixed function approximator. Our method applies to any episodic MDP, but it works best in domains—such as resource-constrained scheduling and other combinatorial optimization problems— that are discrete and deterministic.

## 2 Preliminaries

There are two distinct reasons for studying value function approximation methods. The primary reason is to be able to generalize from some set of training experiences to produce a policy that can be applied in new states that were not visited during training. For example, in Tesauro's (1995) work on backgammon, even after training on 200,000 games, the TD-gammon system needed to be able to generalize to new board positions that it had not previously visited. Similarly, in Zhang's (1995) work on space shuttle scheduling, each individual scheduling problem visits only a finite number of states, but the goal is to learn from a series of "training" problems and generalize to new states that arise in "test" problems. Similar MDPs have been studied by Moll, Barto, Perkins & Sutton (1999).

The second reason to study function approximation is to support learning in continuous state spaces. Consider a robot with sensors that return continuous values. Even during training, it is unlikely that the same vector of sensor readings will ever be experienced more than once. Hence, generalization is critical during the learning process as well as after learning.

The methods described in this paper address only the first of these reason. Specifically, we study the problem of generalizing from a partial policy to construct a complete policy for a Markov Decision Problem (MDP). Formally, consider a discrete time MDP $M$ with probability transition function $P(s'|s,a)$ (probability that state $s'$ will result from executing action $a$ in state $s$) and expected reward function $R(s'|s,a)$ (expected reward received from executing action $a$ in state $s$ and entering state $s'$). We will assume that, as in backgammon and space shuttle scheduling, $P(s'|s,a)$ and $R(s'|s,a)$ are known and available to the agent, but that the state space is so large that it prevents methods such as value iteration or policy iteration from being applied. Let $L$ be a set of "training" states for which we have an approximation $\hat{V}(s)$ to the optimal value function $V^*(s), s \in L$. In some cases, we will also assume the availability of a policy $\hat{\pi}$ consistent with $\hat{V}(s)$. The goal is to construct a parameterized approximation $\tilde{V}(s; \Theta)$ that can be applied to all states in $M$ to yield a good policy $\tilde{\pi}$ via one-step lookahead search. In the experiments reported below, the set $L$ contains states that lie along trajectories from a small set of "training" starting states $S_0$ to terminal states. A successful learning method will be able to generalize to give a good policy for new starting states not in $S_0$. This was the situation that arose in space shuttle scheduling, where the set $L$ contained states that were visited while solving "training" problems and the learned value function was applied to solve "test" problems.

To represent states for function approximation, let $X(s)$ denote a vector of features describing the state $s$. Let $K(X_1, X_2)$ be a kernel function (generalized inner product) of the two feature vectors $X_1$ and $X_2$. In our experiments, we have employed the gaussian kernel: $K(X_1, X_2; \sigma) = \exp(-\|X_1 - X_2\|^2/\sigma^2)$ with parameter $\sigma$.

## 3 Three LP Formulations of Function Approximation

We now introduce three linear programming formulations of the function approximation problem. We first express each of these formulations in terms of a generic fitted function approximator $\tilde{V}$. Then, we implement $\tilde{V}(s)$ as the dot product of a weight vector $W$ with the feature vector $X(s)$: $\tilde{V}(s) = W \cdot X(s)$. Finally, we apply the "kernel trick" by first rewriting $W$ as a weighted sum of the training points $s_j \in L$, $W = \sum_j \alpha_j X(s_j)$, $(\alpha_j \geq 0)$, and then replacing all dot products between data points by invocations of the kernel function $K$. We assume $L$ con-

tains all states along the best paths from $S_0$ to terminal states and also all states that can be reached from these paths in one step and that have been visited during exploration (so that $\hat{V}$ is known). In all three formulations we have employed linear objective functions, but quadratic objectives like those employed in standard support vector machines could be used instead. All slack variables in these formulations are constrained to be non-negative.

**Formulation 1: Supervised Learning.** The first formulation treats the value function approximation problem as a supervised learning problem and applies the standard $\epsilon$-insensitive loss function (Vapnik, 2000) to fit the function approximator.

$$\text{minimize} \quad \sum_s [u(s) + v(s)]$$

$$\text{subject to} \quad \tilde{V}(s) + u(s) \geq \hat{V}(s) - \epsilon; \quad \tilde{V}(s) - v(s) \leq \hat{V}(s) + \epsilon \quad \forall s \in L$$

In this formulation, $u(s)$ and $v(s)$ are slack variables that are non-zero only if $\tilde{V}(s)$ has an absolute deviation from $\hat{V}(s)$ of more than $\epsilon$. The objective function seeks to minimize these absolute deviation errors. A key idea of support vector methods is to combine this objective function with a penalty on the norm of the weight vector. We can write this as

$$\text{minimize} \quad \|W\|_1 + C \sum_s [u(s) + v(s)]$$

$$\text{subject to} \quad W \cdot X(s) + u(s) \geq \hat{V}(s) - \epsilon; \quad W \cdot X(s) - v(s) \leq \hat{V}(s) + \epsilon \quad \forall s \in L$$

The parameter $C$ expresses the tradeoff between fitting the data (by driving the slack variables to zero) and minimizing the norm of the weight vector. We have chosen to minimize the 1-norm of the weight vector ($\|W\|_1 = \sum_i |w_i|$), because this is easy to implement via linear programming. Of course, if the squared Euclidean norm of $W$ is preferred, then quadratic programming methods could be applied to minimize this.

Next, we introduce the assumption that $W$ can be written as a weighted sum of the data points themselves. Substituting this into the constraint equations, we obtain

$$\text{minimize} \quad \sum_j \alpha_j + C \sum_s [u(s) + v(s)]$$

$$\text{subject to} \quad \sum_j \alpha_j X(s_j) \cdot X(s) + u(s) \geq \hat{V}(s) - \epsilon \quad \forall s \in L$$
$$\sum_j \alpha_j X(s_j) \cdot X(s) - v(s) \leq \hat{V}(s) + \epsilon \quad \forall s \in L$$

Finally, we can apply the kernel trick by replacing each dot product by a call to a kernel function:

$$\text{minimize} \quad \sum_j \alpha_j + C \sum_s [u(s) + v(s)]$$

$$\text{subject to} \quad \sum_j \alpha_j K(X(s_j), X(s)) + u(s) \geq \hat{V}(s) - \epsilon \quad \forall s \in L$$
$$\sum_j \alpha_j K(X(s_j), X(s)) - v(s) \leq \hat{V}(s) + \epsilon \quad \forall s \in L$$

**Formulation 2: Bellman Learning.** The second formulation introduces constraints from the Bellman equation $V(s) = \max_a \sum_{s'} P(s'|s,a)[R(s'|s,a) + V(s')]$. The standard approach to solving MDPs via linear programming is the following. For each state $s$ and action $a$,

$$\text{minimize} \quad \sum_{s,a} u(s,a)$$

subject to $\quad V(s) = u(s,a) + \sum_{s'} P(s'|s,a)[R(s'|s,a) + V(s')]$

The idea is that for the optimal action $a^*$ in state $s$, the slack variable $u(s,a^*)$ can be driven to zero, while for non-optimal actions $a_-$, the slack $u(s,a_-)$ will remain non-zero. Hence, the minimization of the slack variables implements the maximization operation of the Bellman equation.

We attempted to apply this formulation with function approximation, but the errors introduced by the approximation make the linear program infeasible, because $\tilde{V}(s)$ must sometimes be less than the backed-up value $\sum_{s'} P(s'|s,a)[R(s'|s,a) + \tilde{V}(s')]$. This led us to the following formulation in which we exploit the approximate value function $\hat{V}$ to provide "advice" to the LP optimizer about which constraints should be tight and which ones should be loose. Consider a state $s$ in $L$. We can group the actions available in $s$ into three groups: (a) the "optimal" action $a^* = \hat{\pi}(s)$ chosen by the approximate policy $\hat{\pi}$, (b) other actions that are tied for optimum (denoted by $a_0$), and (c) actions that are sub-optimal (denoted by $a_-$). We have three different constraint equations, one for each type of action:

$$\text{minimize} \quad \sum_{s}[u(s,a^*) + v(s,a^*)] + \sum_{s,a_0} y(s,a_0) + \sum_{s,a_-} z(s,a_-)$$

$$\text{subject to} \quad \tilde{V}(s) + u(s,a^*) - v(s,a^*) = \sum_{s'} P(s'|s,a^*)[R(s'|s,a^*) + \tilde{V}(s')]$$

$$\tilde{V}(s) + y(s,a_0) \geq \sum_{s'} P(s'|s,a_0)[R(s'|s,a_0) + \tilde{V}(s')]$$

$$\tilde{V}(s) + z(s,a_-) \geq \sum_{s'} P(s'|s,a_-)[R(s'|s,a_-) + \tilde{V}(s')] + \epsilon$$

The first constraint requires $\tilde{V}(s)$ to be approximately equal to the backed-up value of the chosen optimal action $a^*$. The second constraint requires $\tilde{V}(s)$ to be at least as large as the backed-up value of any alternative optimal actions $a_0$. If $\tilde{V}(s)$ is too small, it will be penalized, because the slack variable $y(s,a_0)$ will be non-zero. But there is no penalty if $\tilde{V}(s)$ is too large. The main effect of this constraint is to drive the value of $\tilde{V}(s')$ downward as necessary to satisfy the first constraint on $a^*$. Finally, the third constraint requires that $\tilde{V}(s)$ be at least $\epsilon$ larger than the backed-up value of all inferior actions $a_-$. If these constraints can be satisfied with all slack variables $u, v, y$, and $z$ set to zero, then $\tilde{V}$ satisfies the Bellman equation.

After applying the kernel trick and introducing the regularization objective, we obtain the following Bellman formulation:

$$\text{minimize} \quad \sum_{j} \alpha_j + C \left( \sum_{s,a_0,a_-} u(s,a^*) + v(s,a^*) + y(s,a_0) + z(s,a_-) \right)$$

$$\text{subject to} \quad \sum_{j} \alpha_j \left[ K(X(s_j), X(s)) - \sum_{s'} P(s'|s,a^*) K(X(s_j), X(s')) \right] +$$

$$u(s,a^*) - v(s,a^*) = \sum_{s'} P(s'|s,a^*) R(s'|s,a^*)$$

$$\sum_{j} \alpha_j \left[ K(X(s_j), X(s)) - \sum_{s'} P(s'|s,a_0) K(X(s_j), X(s')) \right] + y(s,a_0)$$

$$\geq \sum_{s'} P(s'|s,a_0) R(s'|s,a_0)$$

$$\sum_j \alpha_j \left[ K(X(s_j), X(s)) - \sum_{s'} P(s'|s, a_-) K(X(s_j), X(s')) \right] + z(s, a_-)$$
$$\geq \sum_{s'} P(s'|s, a_-) R(s'|s, a_-) + \epsilon$$

**Formulation 3: Advantage Learning.** The third formulation focuses on the minimal constraints that must be satisfied to ensure that the greedy policy computed from $\tilde{V}$ will be identical to the greedy policy computed from $\hat{V}$ (cf. Utgoff & Saxena, 1987). Specifically, we require that the backed up value of the optimal action $a^*$ be greater than the backed up values of all other actions $a$.

minimize $\quad \displaystyle\sum_{s, a^*, a} u(s, a^*, a)$

subject to $\quad \displaystyle\sum_{s'} P(s'|s, a^*)[R(s'|s, a^*) + \tilde{V}(s')] + u(s, a^*, a)$
$$\geq \sum_{s'} P(s'|s, a)[R(s'|s, a) + \tilde{V}(s')] + \epsilon$$

There is one constraint and one slack variable $u(s, a^*, a)$ for every action executable in state $s$ except for the chosen optimal action $a^* = \hat{\pi}(s)$. The backed-up value of $a^*$ must have an advantage of at least $\epsilon$ over any other action $a$, even other actions that, according to $\hat{V}$, are just as good as $a^*$. After applying the kernel trick and incorporating the complexity penalty, this becomes

minimize $\quad \displaystyle\sum_j \alpha_j + C \sum_{s, a^*, a} u(s, a^*, a)$

subject to $\quad \displaystyle\sum_j \alpha_j \sum_{s'} [P(s'|s, a^*) - P(s'|s, a)] K(X(s_j), X(s')) + u(s, a^*, a) \geq$
$$\sum_{s'} P(s'|s, a) R(s'|s, a) - \sum_{s'} P(s'|s, a^*) R(s'|s, a^*) + \epsilon$$

Of course each of these formulations can easily be modified to incorporate a discount factor for discounted cumulative reward.

## 4    Experimental Results

To compare these three formulations, we generated a set of 10 random maze problems as follows. In a 100 by 100 maze, the agent starts in a randomly-chosen square in the left column, $(0, y)$. Three actions are available in every state, east, northeast, and southeast, which deterministically move the agent one square in the indicated direction. The maze is filled with 3000 rewards (each of value $-5$) generated randomly from a mixture of a uniform distribution (with probability 0.20) and five 2-D gaussians (each with probability 0.16) centered at (80,20), (80,60), (40,20), (40,80), and (20,50) with variance 10 in each dimension. Multiple rewards generated for a single state are accumulated. In addition, in column 99, terminal rewards are generated according to a distribution that varies from $-5$ to $+15$ with minima at (99,0), (99,40), and (99,80) and maxima at (99,20) and (99,60).

Figure 1 shows one of the generated mazes. These maze problems are surprisingly hard because unlike "traditional" mazes, they contain no walls. In traditional mazes, the walls tend to guide the agent to the goal states by reducing what would be a 2-D random walk to a random walk of lower dimension (e.g., 1-D along narrow halls).

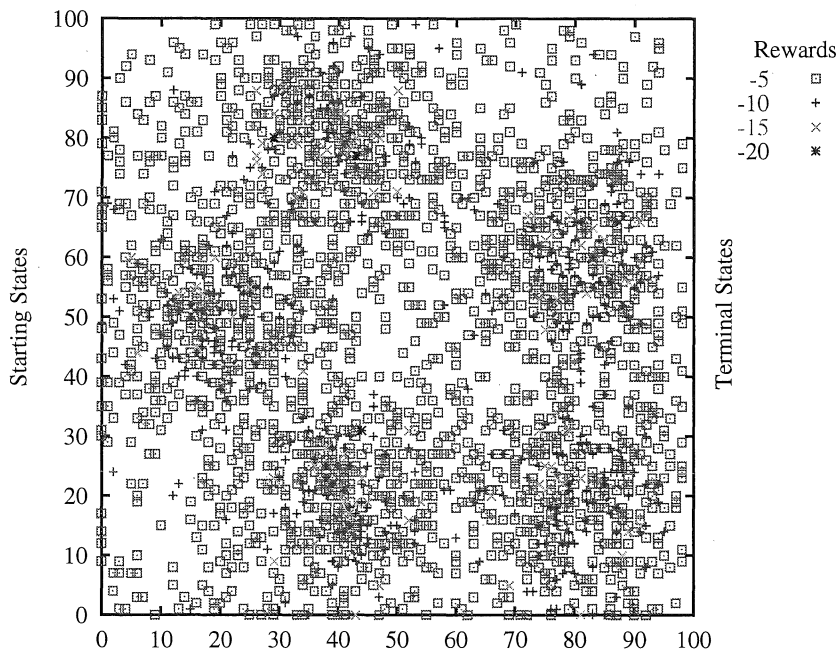

Figure 1: Example randomly-generated maze. Agent enters at left edge and exits at right edge.

We applied the three LP formulations in an incremental-batch method as shown in Table 1. The LPs were solved using the CPLEX package from ILOG. The $\tilde{V}$ giving the best performance on the starting states in $S_0$ over the 20 iterations was saved and evaluated over all 100 possible starting states to obtain a measure of generalization. The values of $C$ and $\sigma$ were determined by evaluating generalization on a holdout set of 3 start states: $(0,30)$, $(0,50)$, and $(0,70)$. Experimentation showed that $C = 100,000$ worked well for all three methods. We tuned $\sigma^2$ separately for each problem using values of 5, 10, 20, 40, 60, 80, 120, and 160; larger values were preferred in case of ties, since they give better generalization. The results are summarized in Figure 2.

The figure shows that the three methods give essentially identical performance, and that after 3 examples, all three methods have a regret per start state of about 2 units, which is less than the cost of a single $-5$ penalty. However, the three formulations differ in their ease of training and in the information they require. Table 2 compares training performance in terms of (a) the CPU time required for training, (b) the number of support vectors constructed, (c) the number of states in which $\tilde{V}$ prefers a tied-optimal action over the action chosen by $\hat{\pi}$, (d) the number of states in which $\tilde{V}$ prefers an inferior action, and (e) the number of iterations performed *after* the best-performing iteration on the training set. A high score on this last measure indicates that the learning algorithm is not converging well, even though it may momentarily attain a good fit to the data. By virtually every measure, the advantage formulation scores better. It requires much less CPU time to train, finds substantially fewer support vectors, finds function approximators that give better fit to the data, and tends to converge better. In addition, the advantage

Table 1: Incremental Batch Reinforcement Learning

Repeat 20 times:
    For each start state $s_0 \in S_0$ do
        Generate 16 $\epsilon$-greedy trajectories using $\bar{V}$
        Record all transitions and rewards to build MDP model $\hat{M}$
    Solve $\hat{M}$ via value iteration to obtain $\hat{V}$ and $\hat{\pi}$
    $L = \emptyset$
    For each start state $s_0 \in S_0$ do
        Generate trajectory according to $\hat{\pi}$
        Add to $L$ all states visited along this trajectory
    Apply LP method to $L$, $\hat{V}$, and $\hat{\pi}$ to find new $\tilde{V}$
Perform Monte Carlo rollouts using greedy policy for $\tilde{V}$ to evaluate each possible start state
Report total value of all start states.

Table 2: Measures of the quality of the training process (average over 10 MDPs)

| | $\|S_0\| = 1$ | | | | | $\|S_0\| = 2$ | | | | |
| | CPU | #SV | #tie | #bad | #iter | CPU | #SV | #tie | #bad | #iter |
| --- | --- | --- | --- | --- | --- | --- | --- | --- | --- | --- |
| Sup | 37.5 | 29.5 | 22.4 | 0.7 | 5.6 | 190.7 | 54.3 | 49.8 | 1.9 | 7.3 |
| Bel | 30.4 | 40.9 | 18.8 | 0.9 | 5.9 | 92.7 | 51.1 | 47.9 | 0.4 | 8.2 |
| Adv | 11.7 | 17.2 | 19.4 | 0.2 | 1.6 | 38.4 | 39.6 | 29.1 | 1.4 | 2.0 |

| | $\|S_0\| = 3$ | | | | | $\|S_0\| = 4$ | | | | |
| | CPU | #SV | #tie | #bad | #iter | CPU | #SV | #tie | #bad | #iter |
| --- | --- | --- | --- | --- | --- | --- | --- | --- | --- | --- |
| Sup | 433.2 | 105.5 | 70.5 | 3.0 | 10.5 | 789.1 | 117.2 | 90.5 | 3.3 | 9.6 |
| Bel | 208.0 | 82.4 | 62.0 | 2.2 | 3.3 | 379.1 | 145.7 | 75.2 | 1.8 | 7.3 |
| Adv | 74.5 | 58.6 | 46.7 | 0.6 | 4.0 | 122.4 | 74.0 | 51.9 | 3.2 | 2.8 |

and Bellman formulations do not require the value of $\hat{V}$, but only $\hat{\pi}$. This makes them suitable for learning to imitate a human-supplied policy.

## 5 Conclusions

This paper has presented three formulations of batch value function approximation by exploiting the power of linear programming to express a variety of constraints and borrowing the kernel trick from support vector machines. All three formulations were able to learn and generalize well on difficult synthetic maze problems. The advantage formulation is easier and more reliable to train, probably because it places fewer constraints on the value function approximation. Hence, we are now applying the advantage formulation to combinatorial optimization problems in scheduling and protein structure determination.

### Acknowledgments

The authors gratefully acknowledge the support of AFOSR under contract F49620-98-1-0375, and the NSF under grants IRI-9626584, IIS-0083292, ITR-5710001197, and EIA-9818414. We thank Valentina Zubek and Adam Ashenfelter for their careful reading of the paper.

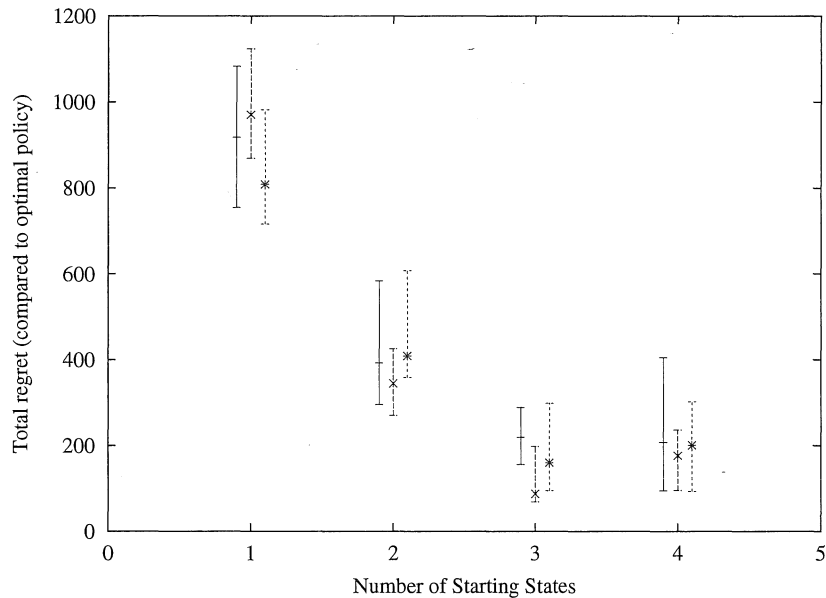

Figure 2: Comparison of the total regret (optimal total reward − attained total reward) summed over all 100 starting states for the three formulations as a function of the number of start states in $S_0$. The three error bars represent the performance of the supervised, Bellman, and advantage formulations (left-to-right). The bars plot the 25th, 50th, and 75th percentiles computed over 10 randomly generated mazes. Average optimal total reward on these problems is 1306. The random policy receives a total reward of −14, 475.

# References

Baird, L. C. (1993). Advantage updating. Tech. rep. 93-1146, Wright-Patterson AFB.

Moll, R., Barto, A. G., Perkins, T. J., & Sutton, R. S. (1999). Learning instance-independent value functions to enhance local search. NIPS-11, 1017–1023.

Tesauro, G. (1995). Temporal difference learning and TD-Gammon. *CACM*, *28*(3), 58–68.

Utgoff, P. E., & Saxena, S. (1987). Learning a preference predicate. In *ICML-87*, 115–121.

Vapnik, V. (2000). *The Nature of Statistical Learning Theory, 2nd Ed.* Springer.

Zhang, W., & Dietterich, T. G. (1995). A reinforcement learning approach to job-shop scheduling. In *IJCAI95*, 1114–1120.
